# Probabilistic Inference in Human Sensorimotor Processing

**Konrad P. Körding** *
Institute of Neurology
UCL London
London WC1N 3BG,UK
konrad@koerding.com

**Daniel M. Wolpert** †
Institute of Neurology
UCL London
London WC1N 3BG,UK
wolpert@ion.ucl.ac.uk

## Abstract

When we learn a new motor skill, we have to contend with both the variability inherent in our sensors and the task. The sensory uncertainty can be reduced by using information about the distribution of previously experienced tasks. Here we impose a distribution on a novel sensorimotor task and manipulate the variability of the sensory feedback. We show that subjects internally represent both the distribution of the task as well as their sensory uncertainty. Moreover, they combine these two sources of information in a way that is qualitatively predicted by optimal Bayesian processing. We further analyze if the subjects can represent multimodal distributions such as mixtures of Gaussians. The results show that the CNS employs probabilistic models during sensorimotor learning even when the priors are multimodal.

## 1 Introduction

Real world motor tasks are inherently uncertain. For example, when we try to play an approaching tennis ball, our vision of the ball does not provide perfect information about its velocity. Due to this sensory uncertainty we can only generate an estimate of the ball's velocity. This uncertainty can be reduced by taking into account information that is available on a longer time scale: not all velocities are a priori equally probable. For example, very fast and very slow balls may be experienced less often than medium paced balls. Over the course of a match there will be a probability distribution of velocities. Bayesian theory [1-2] tells us that to make an optimal estimate of the velocity of a given ball, this a priori information about the distribution of velocities should be combined with the evidence provided by sensory feedback. This combination process requires prior knowledge, how probable each possible velocity is, and knowledge of the uncertainty inherent in the sensory estimate of velocity. As the degree of uncertainty in the feedback increases, for example when playing in fog or at dusk, an optimal system should increasingly depend on prior knowledge. Here we examine whether subjects represent the probability distribution of a task and if this can be appropriately combined with an estimate of sensory uncer-

tainty. Moreover, we examine whether subjects can represent priors that have multimodal distributions.

## 2   Experiment 1: Gaussian Prior

To examine whether subjects can represent a prior distribution of a task and integrate it with a measure of their sensory uncertainty we examined performance on a reaching task. The perceived position of the hand is displaced relative to the real position of the hand. This displacement or shift is drawn randomly from an underlying probability distribution and subjects have to estimate this shift to perform well on the task. By examining where subjects reached while manipulating the reliability of their visual feedback we distinguished between several models of sensorimotor learning.

### 2.1   Methods

Ten subjects made reaching movement on a table to a visual target with their right index finger in a virtual reality setup (for details of the set-up see [6]). An Optotrak 3020 measured the position of their finger and a projection/mirror system prevented direct view of their arm and allowed us to generate a cursor representing their finger position which was displayed in the plane of the movement (Figure 1A). As the finger moved from the starting circle, the cursor was extinguished and shifted laterally from the true finger location by an amount $x_{true}$ which was drawn each trial from a Gaussian distribution:

$$p(x_{true}) = \frac{1}{\sqrt{2\pi}\sigma_{prior}}e^{\frac{(x_{true}-x_{dist})^2}{2\sigma_{prior}^2}}$$   (1)

where $x_{dist} = 1cm$ and $\sigma_{prior} = 0.5cm$ (Figure 1B). Halfway to the target (10 cm), visual feedback was briefly provided for 100 ms either clearly ( $\sigma_0$) or with different degrees of blur ( $\sigma_M$ and $\sigma_L$), or withheld ( $\sigma_\infty$). On each trial one of the 4 types of feedback ($\sigma_0, \sigma_M, \sigma_L, \sigma_\infty$) was selected randomly, with the relative frequencies of (3, 1, 1, 1) respectively. The ($\sigma_0$) feedback was a small white sphere. The ($\sigma_M$) feedback was 25 small translucent spheres, distributed as a 2 dimensional Gaussian with a standard deviation of 1 cm, giving a cloud type impression. The ($\sigma_L$) feedback was analogous but with a standard deviation of 2 cm. No feedback was provided in the ($\sigma_\infty$) case. After another 10 cm of movement the trial finished and feedback of the final cursor location was only provided in the ($\sigma_0$) condition. The experiment consisted of 2000 trials for each subject. Subjects were instructed to take into account what they see at the midpoint and get as close to the target as possible and that the cursor is always there even if it is not displayed.

### 2.2   Results: Trajectories in the Presence of Uncertainty

Subjects were trained for 1000 trials on the task to ensure that they experienced many samples $x_{true}$ drawn from the underlying distribution $p(x_{true})$. After this period, when feedback was withheld ($\sigma_\infty$), subjects pointed 0.97$\pm$ 0.06 cm (mean$\pm$ se across subjects) to the left of the target showing that they had learned the average shift of 1 cm experienced over the trials. Subsequently, we examined the relationship between visual feedback and the location $x_{estimate}$ subjects pointed to. On trials in which feedback was provided, there was compensation during the second half of the movement. Figure 1A shows typical finger and cursor paths for two trials, $\sigma_\infty$ and $\sigma_0$, in which $x_{true} = 2cm$. The visual feedback midway through the movement provides information about the lateral shift on the current trial and allows for a correction for the current lateral shift. However, the visual system is not perfect and we expect some uncertainty in the sensed lateral shift $x_{sensed}$. The distribution of sensed shifts over a large number of trials is expected to have a Gaussian

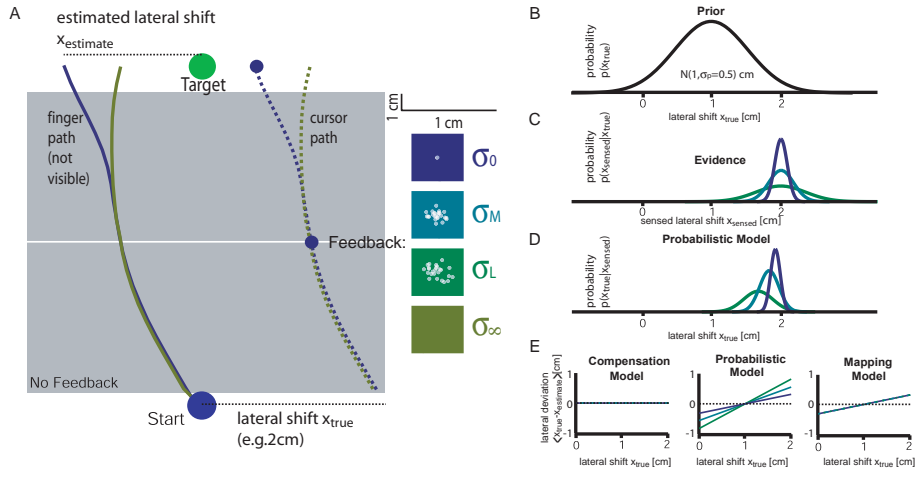

Figure 1: The experiment and models. A) Subjects are required to place the cursor on the target, thereby compensating for the lateral displacement. The finger paths illustrate typical trajectories at the end of the experiment when the lateral shift was 2 cm (the colors correspond to two of the feedback conditions). B) The experimentally imposed prior distribution of lateral shifts is Gaussian with a mean of 1 cm. C) A schematic of the probability distribution of visually sensed shifts under clear and the two blurred feedback conditions (colors as in panel A) for a trial in which the true lateral shift is 2 cm. D) The estimate of the lateral shift for an optimal observer that combines the prior with the evidence. E) The average lateral deviation from the target as a function of the true lateral shift for the models. Left: the full compensation model. Middle the Bayesian probabilistic model. Right: the mapping model (see text for details).

distribution centered on $x_{true}$ with a standard deviation $\sigma_{sensed}$ that depends on the acuity of the system.

$$p(x_{sensed}|x_{true}) = \frac{1}{\sqrt{2\pi}\sigma_{sensed}}e^{\frac{(x_{true}-x_{sensed})^2}{2\sigma_{sensed}^2}} \tag{2}$$

As the blur increases we expect $\sigma_{sensed}$ to increase (Figure 1C).

## 2.3 Computational Models and Predictions

There are several computational models which subjects could use to determine the compensation needed to reach the target based on the sensed location of the finger midway through the movement. To analyze the subjects performance we plot the average lateral deviation $\langle x_{true} - x_{estimated}\rangle$ in a set of bins of as a function of the true shift $x_{true}$. Because feedback is not biased this term approximates $\langle x_{sensed} - x_{estimated}\rangle$. Three competing computational models are able to predict such a graph.

1) Compensation model. Subjects could compensate for the sensed lateral shift $x_{sensed}$ and thus use $x_{estimated} = x_{sensed}$. The average lateral deviation should thus be $\langle x_{true} - x_{estimated}\rangle = 0$ (Figure 1E, left panel). In this model, increasing the uncertainty of the feedback $\sigma_{sensed}$ (by increasing the blur) affects the variability of the pointing but not the average location. Errors arise from variability in the visual feedback and the means squared error (MSE) for this strategy (ignoring motor variability) is $\sigma_{sensed}^2$. Crucially this model does not require subjects to estimate their visual uncertainty nor the distribution of

shifts.

2) Bayesian model. Subjects could optimally use prior information about the distribution and the uncertainty of the visual feedback to estimate the lateral shift. They have to estimate $x_{true}$ given $x_{sensed}$. Using Bayes rule we can obtain the posterior distribution, that is the probability of a shift $x_{true}$ given the evidence $x_{sensed}$,

$$p(x_{true}|x_{sensed}) = \frac{p(x_{true})p(x_{sensed}|x_{true})}{p(x_{sensed})} \qquad (3)$$

If subjects choose the most likely shift they also minimize their mean squared error (MSE). We can determine this optimal estimate $x_{estimated}$ by differentiating (3) after inserting (1) and (2). This optimal estimate is a weighted sum between the mean of the prior and the sensed feedback position:

$$x_{estimated} = \frac{\sigma_{sensed}^2}{\sigma_{sensed}^2 + \sigma_{prior}^2}x_{dist} + \frac{\sigma_{prior}^2}{\sigma_{sensed}^2 + \sigma_{prior}^2}x_{sensed} \qquad (4)$$

The average lateral deviation $\langle x_{true} - x_{estimated} \rangle$ is thus linearly dependent to $x_{true}$ and the slope increases with increasing uncertainty (Figure 1E middle panel).

The MSE depends on two factors, the width of the prior $\sigma_{prior}$ and the uncertainty in the visual feedback $\sigma_{sensed}$. Calculating the MSE for the above optimal choice we obtain:

$$MSE = \frac{\sigma_{prior}^2}{\sigma_{prior}^2 + \sigma_{sensed}^2}\sigma_{sensed}^2 \qquad (5)$$

which is always less than the MSE for model 1. As we increase the blur, and thus the degree of uncertainty, the estimate of the shift moves away from the visually sensed displacement $x_{sensed}$ towards the mean of the prior distribution $x_{dist}$ (Figure 1D). Such a computational strategy thus allows subjects to minimize the MSE at the target.

3) Mapping model. A third computational strategy is to learn a mapping from the sensed shift $x_{sensed}$ to the optimal lateral shift $x_{estimated}$. By minimizing the average error over many trials the subjects could achieve a combination similar to model 2 but without any representation of the prior distribution or the visual uncertainty. However, to learn such a mapping requires visual feedback and knowledge of the error at the end of the movement. In our experiment we only revealed the shifted position of the finger at the end of the movement of the clear feedback trials ($\sigma_0$). Therefore, if subjects learn a mapping, they can only do so for these trials and apply the same mapping to the blurred conditions ($\sigma_M$, $\sigma_L$). Therefore, this model predicts that the average lateral shift $\langle x_{true} - x_{estimated} \rangle$ should be independent of the degree of blur (Figure 1E right panel)

### 2.3.1 Results: Lateral Deviation

Graphs of $\langle x_{true} - x_{estimated} \rangle$ against $x_{true}$ are shown for a representative subject in Figure 2A. The slope increases with increasing uncertainty and is, therefore, incompatible with models 1 and 3 but is predicted by model 2. Moreover, this transition from using feedback to using prior information occurs gradually with increasing uncertainty as also predicted by this Bayesian model. These effects are consistent over all the subjects tested. The slope increases with increasing uncertainty in the visual feedback (Figure 2B). Depending on the uncertainty of the feedback, subjects thus combine prior knowledge of the distribution of shifts with new evidence to generate the optimal compensatory movement.

Using Bayesian theory we can furthermore infer the degree of uncertainty from the errors the subjects made. Given the width of the prior $\sigma_{prior} = 0.5cm$ and the result in (4) we can

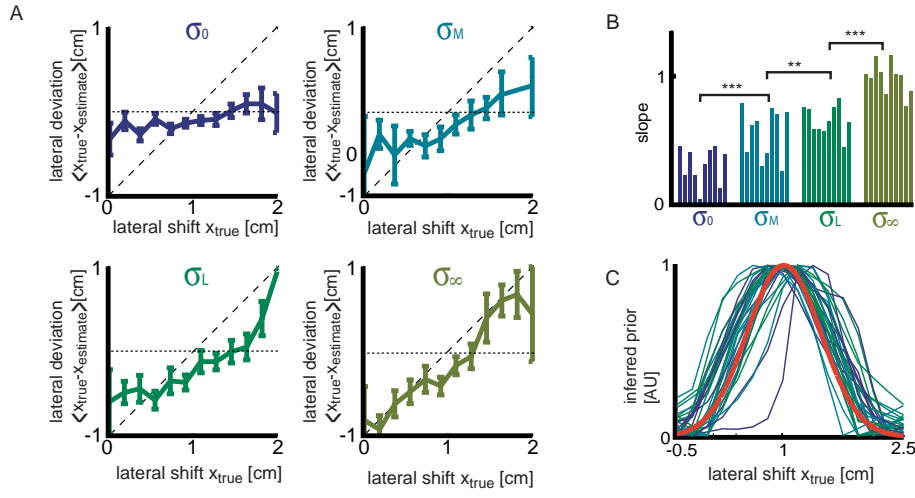

Figure 2: Results with color codes as in Figure 1. A) The average lateral deviation of the cursor at the end of the trial as a function of the imposed lateral shift for a typical subject. Errorbars denote the s.e.m. The horizontal dotted lines indicate the prediction from the full compensation model and sloped line for a model that ignores sensory feedback on the current trial and corrects only for the mean over all trials. B) The slopes for the optimal linear fits are shown for the full population of subjects. The stars indicate the significance indicated by the paired t-test. C) The inferred priors and the real prior (red) for each subjects and condition.

estimate the uncertainty $\sigma_{sensed}$ from Fig 2A. For the three levels of imposed uncertainty, $\sigma_0$, $\sigma_M$ and $\sigma_L$, we find that the subjects uncertainty $\sigma_{sensed}$ are $0.36\pm0.1$, $0.67\pm0.3$, $0.8\pm0.2$ cm (mean$\pm$sd across subjects), respectively. Furthermore we have developed a novel technique to infer the priors used by the subjects. An obvious choice of $x_{estimate}$ is the maximum of the posterior $p(x_{true}|x_{sensed})$. The derivative of this posterior with respect to $x_{true}$ must vanish at the optimal $x_{estimate}$. This allows us to estimate the prior used by each subject. Taking derivatives of (3) after inserting (2) and setting to zero we get:

$$\frac{dp(x_{true})}{dx_{true}} \frac{1}{p(x_{true})}\bigg|_{x_{estimate}} = \frac{x_{sensed} - x_{estimated}}{\sigma^2_{sensed}} \tag{6}$$

We assume that $x_{estimate}$ has a narrow peak around $x_{true}$ and thus approximate it by $x_{true}$. We insert the $\sigma_{sensed}$ obtained in (4), affecting the scaling of the integral but not its form. The average of $x_{sensed}$ across many trials is the imposed shift $x_{true}$. Therefore the right hand side is measured in the experiment and the left hand side approximates the derivative of $\log p(x_{true})$. Since $p$ must approach zero for both very small and very large $x_{true}$, we subtract the mean of the right hand side before integrating numerically to obtain an estimate the prior $p(x_{true})$. Figure 2C shows the priors inferred for each subject and condition. This shows that the real prior (red line) was reliably learned by each subject.

## 3   Experiment 2: Mixture of Gaussians Priors

The second experiment was designed to examine whether subjects are able to represent more complicated priors such as mixtures of Gaussians and if they can utilize such prior knowledge.

### 3.1 Methods

12 additional subjects participated in an experiment similar to Experiment 1 with the following changes. The experiments lasted for twice as many trials run on two consecutive days with 2000 trials performed on each day. Feedback midway through the movement was always blurred (spheres distributed as a two dimensional Gaussian with $\sigma = 4cm$) and feedback at the end of the movement was provided on every trial. The prior distribution was a mixture of Gaussians ( Figure 3A,D). One group of 6 subjects was exposed to:

$$p(x_{true}) = \frac{1}{2\sqrt{2\pi}\sigma_{prior}} \left( e^{\frac{(x_{true} - x_{dist})^2}{2\sigma_{prior}^2}} + e^{\frac{(x_{true} + x_{dist})^2}{2\sigma_{prior}^2}} \right) \tag{7}$$

where $x_{dist} = 2cm$ is half the distance between the two peaks of the Gaussians. $\sigma_{prior}$ is the width of each Gaussian which is set to 0.5 cm. Another group of 6 subjects experienced

$$p(x_{true}) = \frac{1}{10\sqrt{2\pi}\sigma_{prior}} \left( e^{\frac{(x_{true} - x_{dist})^2}{2\sigma_{prior}^2}} + e^{\frac{(x_{true} + x_{dist})^2}{2\sigma_{prior}^2}} + 8e^{\frac{x^2}{2\sigma_{prior}^2}} \right) \tag{8}$$

In this case we set $x_{dist} = 2\sqrt{5}$ so that the variance is identical to the two Gaussians case. $\sigma_{prior}$ is still 0.5 cm.

To estimate the priors learned by the subjects we fitted and compared two models. The first assumed that subjects learned a single Gaussian distribution and the second assumed that subjects learned a mixture of Gaussians and we tuned the position of the Gaussians to minimizes the MSE between predicted and actual data.

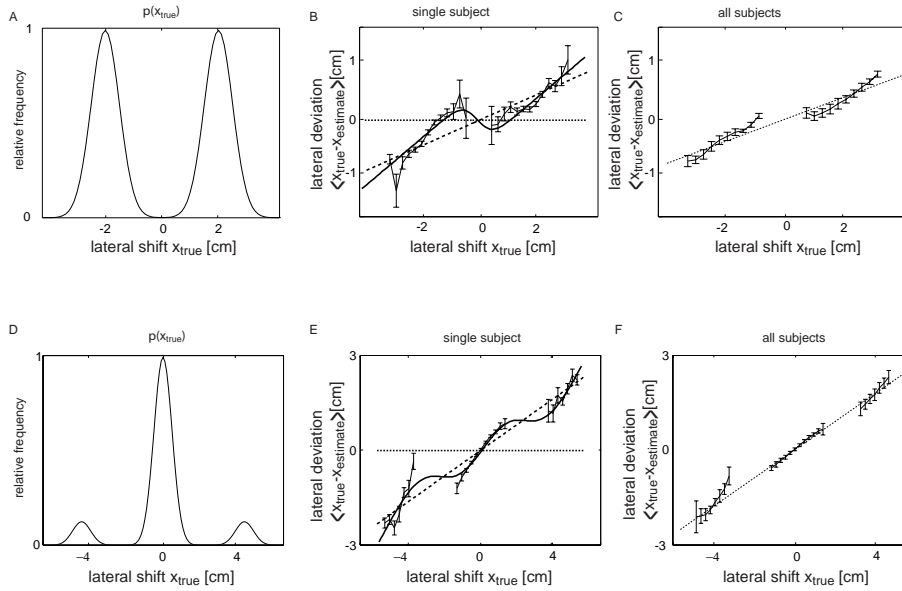

Figure 3: A) The used distribution of $x_{true}$ as mixture of Gaussians model. B) The performance of an arbitrarily chosen subject is shown together with a fit from the ignore prior model (dotted line), the Gaussian model (dashed line) and the Bayesian Mixture of Gaussians model (solid line) C) the average response over all subjects is shown D-F) shows the same as A-C) for the Three Gaussian Distribution

### 3.2   Results: Two Gaussians Distribution

The resulting response graphs (Figure 3B,C) show clear nonlinear effects. Fitting the $\sigma_{sensed}$ and $x_{dist}$ to a two component Mixture of Gaussians model led to an average error over all 6 subjects of 0.14±0.01 cm compared to an average error obtained for a single Gaussian of 0.19±0.02 cm for the two Gaussians model. The difference, is significant at $p < 0.01$. The mixture model of the prior is thus better able to explain the data than the model that assumes that people can just represent one Gaussian. One of the subjects compensated least for the feedback and his data was well fit by a single Gaussian. After removing this subject from the dataset we could fit the width of the distribution $x_{dist}$ and obtained 2.4±0.4 cm, close to the real value of the probability density function of 2 cm.

### 3.3   Results: Three Gaussians Distribution

The resulting response graphs (Figure 3E,F) again show clear nonlinear effects. Fitting the $\sigma_{sensed}$ and $x_{dist}$ of the three Gaussians model (Figure 3E) led to an average error over all subjects of 0.21±0.02 cm instead of an error from a single Gaussians of 0.25±0.02 cm. The fitted distance $x_{dist}$ however was 2.0±0.4 cm, significantly smaller than the real distance.

This result shows that subjects can not fully learn this more complicated distribution but rather just learn some of its properties. This could be due to several effects. First, large values of $x_{true}$ are experienced only rarely. Second, it could be that subjects use a simpler model such as a generalized Gaussian (the family of distribution that also the Laplacian distribution belongs to) or that they use a mixture of only a few Gaussians model. Third, subjects could have a prior over priors that makes a mixture of three Gaussians model very unlikely. Learning such a mixture would therefore be expected to take far longer.

### 3.4   Results: Evolution of the Subjects Performance

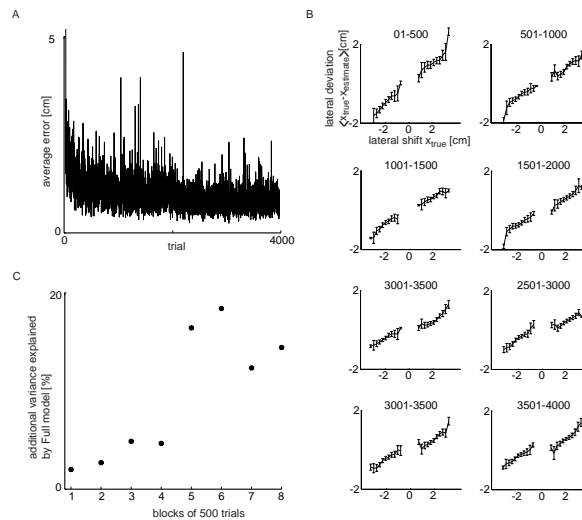

Figure 4: A) The mean error over the 6 subjects is shown as a function of the trial number B) The average lateral deviation as a function of the shift and the trial number C) The additional variance explained by the full model is plotted as a function of the trial number

As a next step we wanted to analyze how the behaviour of the subjects changes over the

course of training. During the process of training the average error over batches of 500 subsequent trials decreased from 1.97 cm to 0.84 cm (Figure 4A). What change leads to this decrease?

To address this we plot the evolution of the lateral deviation graph, as a function of the trial number (Figure 4B). Subjects initially exhibit a slope of about 1 and approximately linear behaviour. This indicates that initially they are using a narrow Gaussian prior. In other words they rely on the prior belief that their hand will not be displaced and ignore the feedback. Only later during training do they show behaviour that is consistent with a bimodal Gaussians distribution.

In Figure 4C we plot the percentage of additional variance explained by the full model when compared to the Gaussian model averaged over the population. It seems that in particular after trial 2000, the trial after which people enjoy a nights rest, does the explanatory power of the full model improve. It could be that subjects need a consolidation period to adequately learn the distribution. Such improvements in learning contingent upon sleep have also been observed in visual learning [7].

## 4 Conclusion

We have shown that a prior is used by humans to determine appropriate motor commands and that it is combined with an estimate of sensory uncertainty. Such a Bayesian view of sensorimotor learning is consistent with neurophysiological studies that show that the brain represents the degree of uncertainty when estimating rewards [8-10] and with psychophysical studies addressing the timing of movements [11]. Not only do people represent the uncertainty and combine this with prior information, they are also able to represent and utilize complicated nongaussian priors. Optimally using a priori knowledge might be key to winning a tennis match. Tennis professionals spend a great deal of time studying their opponent before playing an important match - ensuring that they start the match with correct a priori knowledge.

### Acknowledgments

We like to thank Zoubin Ghahramani for inspiring discussions and the Wellcome Trust for financial support. We also like to thank James Ingram for technical support.

## Footnotes

* www.koerding.com

† www.wolpertlab.com

### References

[1] Cox, R.T. (1946) *American Journal of Physics* 17, 1

[2] Bernardo, J.M. & Smith, A.F.M. (1994) Bayesian theory. John Wiley

[3] Berrou, C., Glavieux, A. & Thitimajshima, P. (1993) *Proc. ICC'93 Geneva, Switzerland* 1064

[4] Simoncelli, E.P. & Adelson, E.H. (1996) *Proc. 3rd International Conference on Image Processing Lausanne, Switzerland*

[5] Weiss, Y., Simoncelli, E.P. & Adelson, E.H. (2002) *Nature Neuroscience* 5, 598

[6] Goodbody, W. & Wolpert, D. (1998) *Journal of Neurophysiology* 79,1825

[7] Stickgold, R., James, L. & Hobson, J.A. (2000) *Nature* 3 ,1237

[8] Fiorillo, C.D., Tobler, P.N. & Schultz, W. (2003) *Science* 299, 1898

[9] Basso, M.A. & Wurt, R.H. (1998) *Journal of Neuroscience* 18, 7519

[10] Platt M.L. (1999) *Nature* 400, 233

[11] Carpenter, R.H. & Williams, M.L. *Nature* 377, 59
